# Hyperparameter and Kernel Learning for Graph Based Semi-Supervised Classification

**Ashish Kapoor[†], Yuan (Alan) Qi[‡], Hyungil Ahn[†] and Rosalind W. Picard[†]**
[†]MIT Media Laboratory, Cambridge, MA 02139
{kapoor, hiahn, picard}@media.mit.edu
[‡]MIT CSAIL, Cambridge, MA 02139
alanqi@csail.mit.edu

## Abstract

There have been many graph-based approaches for semi-supervised classification. One problem is that of hyperparameter learning: performance depends greatly on the hyperparameters of the similarity graph, transformation of the graph Laplacian and the noise model. We present a Bayesian framework for learning hyperparameters for graph-based semi-supervised classification. Given some labeled data, which can contain inaccurate labels, we pose the semi-supervised classification as an inference problem over the unknown labels. Expectation Propagation is used for approximate inference and the mean of the posterior is used for classification. The hyperparameters are learned using EM for evidence maximization. We also show that the posterior mean can be written in terms of the kernel matrix, providing a Bayesian classifier to classify new points. Tests on synthetic and real datasets show cases where there are significant improvements in performance over the existing approaches.

## 1   Introduction

A lot of recent work on semi-supervised learning is based on regularization on graphs [5]. The basic idea is to first create a graph with the labeled and unlabeled data points as the vertices and with the edge weights encoding the similarity between the data points. The aim is then to obtain a labeling of the vertices that is both smooth over the graph and compatible with the labeled data. The performance of most of these algorithms depends upon the edge weights of the graph. Often the smoothness constraints on the labels are imposed using a transformation of the graph Laplacian and the parameters of the transformation affect the performance. Further, there might be other parameters in the model, such as parameters to address label noise in the data. Finding a right set of parameters is a challenge, and usually the method of choice is cross-validation, which can be prohibitively expensive for real-world problems and problematic when we have few labeled data points.

Most of the methods ignore the problem of learning hyperparameters that determine the similarity graph and there are only a few approaches that address this problem. Zhu et al. [8] propose learning non-parametric transformation of the graph Laplacians using semidefinite programming. This approach assumes that the similarity graph is already provided; thus, it does not address the learning of edge weights. Other approaches include label

entropy minimization [7] and evidence-maximization using the Laplace approximation [9].

This paper provides a new way to learn the kernel and hyperparameters for graph based semi-supervised classification, while adhering to a Bayesian framework. The semi-supervised classification is posed as a Bayesian inference. We use the evidence to simultaneously tune the hyperparameters that define the structure of the similarity graph, the parameters that determine the transformation of the graph Laplacian, and any other parameters of the model. Closest to our work is Zhu et al. [9], where they proposed a Laplace approximation for learning the edge weights. We use Expectation Propagation (EP), a technique for approximate Bayesian inference that provides better approximations than Laplace. An additional contribution is a new EM algorithm to learn the hyperparameters for the edge weights, the parameters of the transformation of the graph spectrum. More importantly, we explicitly model the level of label noise in the data, while [9] does not do. We provide what may be the first comparison of hyperparameter learning with cross-validation on state-of-the-art algorithms (LLGC [6] and harmonic fields [7]).

## 2  Bayesian Semi-Supervised Learning

We assume that we are given a set of data points $\mathbf{X} = \{\mathbf{x}_1, .., \mathbf{x}_{n+m}\}$, of which $\mathbf{X}_L = \{\mathbf{x}_1, .., \mathbf{x}_n\}$ are labeled as $\mathbf{t}_L = \{t_1, .., t_n\}$ and $\mathbf{X}_U = \{\mathbf{x}_{n+1}, .., \mathbf{x}_{n+m}\}$ are unlabeled. Throughout this paper we limit ourselves to two-way classification, thus $t \in \{-1, 1\}$. Our model assumes that the hard labels $t_i$ depend upon hidden soft-labels $y_i$ for all $i$. Given the dataset $D = [\{\mathbf{X}_L, \mathbf{t}_L\}, \mathbf{X}_U]$, the task of semi-supervised learning is then to infer the posterior $p(\mathbf{t}_U|D)$, where $\mathbf{t}_U = [t_{n+1}, .., t_{n+m}]$. The posterior can be written as:

$$p(\mathbf{t}_U|D) = \int_{\mathbf{y}} p(\mathbf{t}_U|\mathbf{y}) p(\mathbf{y}|D) \tag{1}$$

In this paper, we propose to first approximate the posterior $p(\mathbf{y}|D)$ and then use (1) to classify the unlabeled data. Using the Bayes rule we can write:

$$p(\mathbf{y}|D) = p(\mathbf{y}|\mathbf{X}, \mathbf{t}_L) \propto p(\mathbf{y}|\mathbf{X}) p(\mathbf{t}_L|\mathbf{y})$$

The term, $p(\mathbf{y}|\mathbf{X})$ is the prior. It enforces a smoothness constraint and depends upon the underlying data manifold. Similar to the spirit of graph regularization [5] we use similarity graphs and their transformed Laplacian to induce priors on the soft labels $\mathbf{y}$. The second term, $p(\mathbf{t}_L|\mathbf{y})$ is the likelihood that incorporates the information provided by the labels.

In this paper, $p(\mathbf{y}|D)$ is inferred using Expectation Propagation, a technique for approximate Bayesian inference [3]. In the following subsections first we describe the prior and the likelihood in detail and then we show how evidence maximization can be used to learn hyperparameters and other parameters in the model.

### 2.1  Priors and Regularization on Graphs

The prior plays a significant role in semi-supervised learning, especially when there is only a small amount of labeled data. The prior imposes a smoothness constraint and should be such that it gives higher probability to the labelings that respect the similarity of the graph.

The prior, $p(\mathbf{y}|\mathbf{X})$, is constructed by first forming an undirected graph over the data points. The data points are the nodes of the graph and edge-weights between the nodes are based on similarity. This similarity is usually captured using a kernel. Examples of kernels include RBF, polynomial etc. Given the data points and a kernel, we can construct an $(n+m) \times (n+m)$ kernel matrix $K$, where $K_{ij} = k(\mathbf{x_i}, \mathbf{x_j})$ for all $i \in \{1, .., n+m\}$.

Lets consider the matrix $\tilde{K}$, which is same as the matrix $K$, except that the diagonals are set to zero. Further, if $G$ is a diagonal matrix such that $G_{ii} = \sum_j \tilde{K}_{ij}$, then we can construct the

combinatorial Laplacian ($\Delta = G - \tilde{K}$) or the normalized Laplacian ($\tilde{\Delta} = I - G^{-\frac{1}{2}}\tilde{K}G^{-\frac{1}{2}}$) of the graph. For brevity, in the text we use $\Delta$ as a notation for both the Laplacians. Both the Laplacians are symmetric and positive semidefinite. Consider the eigen decomposition of $\Delta$ where $\{\mathbf{v}_i\}$ denote the eigenvectors and $\{\lambda_i\}$ the corresponding eigenvalues; thus, we can write $\Delta = \sum_{i=1}^{n+m}\lambda_i \mathbf{v}_i \mathbf{v}_i^T$. Usually, a transformation $r(\Delta) = \sum_{i=1}^{n+m} r(\lambda_i)\mathbf{v}_i \mathbf{v}_i^T$ that modifies the spectrum of $\Delta$ is used as a regularizer. Specifically, the smoothness imposed by this regularizer prefers soft labeling for which the norm $\mathbf{y}^T r(\Delta)\mathbf{y}$ is small. Equivalently, we can interpret this probabilistically as following:

$$p(\mathbf{y}|\mathbf{X}) \propto e^{-\frac{1}{2}\mathbf{y}^T r(\Delta)\mathbf{y}} = N(0, r(\Delta)^{-1}) \tag{2}$$

Where $r(\Delta)^{-1}$ denotes the pseudo-inverse if the inverse does not exist. Equation (2) suggests that the labelings with the small value of $\mathbf{y}^T r(\Delta)\mathbf{y}$ are more probable than the others. Note, that when $r(\Delta)$ is not invertible the prior is improper. The fact that the prior can be written as a Gaussian is advantageous as techniques for approximate inference can be easily applied. Also, different choices of transformation functions lead to different semi-supervised learning algorithms. For example, the approach based on Gaussian fields and harmonic functions (Harmonic) [7] can be thought of as using the transformation $r(\lambda) = \lambda$ on the combinatorial Laplacian without any noise model. Similarly, the approach based in local and global consistency (LLGC) [6] can be thought of as using the same transformation but on the normalized Laplacian and a Gaussian likelihood. Therefore, it is easy to see that most of these algorithms can exploit the proposed evidence maximization framework. In the following we focus only on the parametric linear transformation $r(\lambda) = \lambda + \delta$. Note that this transformation removes zero eigenvalues from the spectrum of $\Delta$.

## 2.2 The Likelihood

Assuming conditional independence of the observed labels given the hidden soft labels, the likelihood $p(\mathbf{t}_L|\mathbf{y})$ can be written as $p(\mathbf{t}_L|\mathbf{y}) = \prod_{i=1}^{n} p(t_i|y_i)$. The likelihood models the probabilistic relation between the observed label $t_i$ and the hidden label $y_i$. Many real-world datasets contain hand-labeled data and can often have labeling errors. While most people tend to model label errors with a linear or quadratic slack in the likelihood, it has been noted that such an approach does not address the cases where label errors are far from the decision boundary [2]. The flipping likelihood can handle errors even when they are far from the decision boundary and can be written as:

$$p(t_i|y_i) = \epsilon(1 - \Phi(y_i \cdot t_i)) + (1 - \epsilon)\Phi(y_i \cdot t_i) = \epsilon + (1 - 2\epsilon)\Phi(y_i \cdot t_i) \tag{3}$$

Here, $\Phi$ is the step function, $\epsilon$ is the labeling error rate and the model admits possibility of errors in labeling with a probability $\epsilon$. This likelihood has been earlier used in the context of Gaussian process classification [2][4]. The above described likelihood explicitly models the labeling error rate; thus, the model should be more robust to the presence of label noise in the data. The experiments in this paper use the flipping noise likelihood shown in (3).

## 2.3 Approximate Inference

In this paper, we use EP to obtain a Gaussian approximation of the posterior $p(\mathbf{y}|D)$. Although, the prior derived in section 2.1 is a Gaussian distribution, the exact posterior is not a Gaussian due to the form of the likelihood. We use EP to approximate the posterior as a Gaussian and then equation (1) can be used to classify unlabeled data points. EP has been previously used [3] to train a Bayes Point Machine, where EP starts with a Gaussian prior over the classifiers and produces a Gaussian posterior. Our task is very similar and we use the same algorithm. In our case, EP starts with the prior defined in (2) and incorporates likelihood to approximate the posterior $p(\mathbf{y}|D) \sim N(\bar{\mathbf{y}}, \Sigma_{\mathbf{y}})$.

### 2.4 Hyperparameter Learning

We use evidence maximization to learn the hyperparameters. Denote the parameters of the kernel as $\Theta_K$ and the parameters of transformation of the graph Laplacian as $\Theta_T$. Let $\Theta = \{\Theta_K, \Theta_T, \epsilon\}$, where $\epsilon$ is the noise hyperparameter. The goal is to solve $\hat{\Theta} = \arg\max_\Theta \log[p(\mathbf{t_L}|\mathbf{X}, \Theta)]$.

Non-linear optimization techniques, such as gradient descent or Expectation Maximization (EM) can be used to optimize the evidence. When the parameter space is small then the Matlab function `fminbnd`, based on golden section search and parabolic interpolation, can be used. The main challenge is that the gradient of evidence is not easy to compute.

Previously, an EM algorithm for hyperparameter learning [2] has been derived for Gaussian Process classification. Using similar ideas we can derive an EM algorithm for semi-supervised learning. In the E-step EP is used to infer the posterior $q(\mathbf{y})$ over the soft labels. The M-step consists of maximizing the lower bound:

$$F = \int_\mathbf{y} q(\mathbf{y}) \log \frac{p(\mathbf{y}|\mathbf{X}, \Theta)p(\mathbf{t}_L|\mathbf{y}, \Theta)}{q(\mathbf{y})}$$

$$= -\int_\mathbf{y} q(\mathbf{y}) \log q(\mathbf{y}) + \int_\mathbf{y} q(\mathbf{y}) \log N(\mathbf{y}; 0, r(\Delta)^{-1})$$

$$+ \sum_{i=1}^{n} \int_{y_i} q(y_i) \log\left(\epsilon + (1 - 2\epsilon)\Phi(y_i \cdot t_i)\right) \leq p(\mathbf{t_L}|\mathbf{X}, \Theta)$$

The EM procedure alternates between the E-step and the M-step until convergence.

- **E-Step**: Given the current parameters $\Theta^i$, approximate the posterior $q(\mathbf{y}) \sim N(\bar{\mathbf{y}}, \Sigma_\mathbf{y})$ by EP.
- **M-Step**: Update
$$\Theta^{i+1} = \arg\max_\Theta \int_\mathbf{y} q(\mathbf{y}) \log \frac{p(\mathbf{y}|\mathbf{X}, \Theta)p(\mathbf{t}_L|\mathbf{y}, \Theta)}{q(\mathbf{y})}$$

In the M-step the maximization with respect to the $\Theta$ cannot be computed in a closed form, but can be solved using gradient descent. For maximizing the lower bound, we used gradient based projected BFGS method using Armijo rule and simple line search. When using the linear transformation $r(\lambda) = \lambda + \delta$ on the Laplacian $\Delta$, the prior $p(\mathbf{y}|\mathbf{X}, \Theta)$ can be written as $N(0, (\Delta + \delta I)^{-1})$. Define $\mathbf{Z} = \Delta + \delta I$ then, the gradients of the lower bound with respect to the parameters are as follows:

$$\frac{\partial F}{\partial \Theta_K} = \frac{1}{2}tr(\mathbf{Z}^{-1}\frac{\partial \Delta}{\partial \Theta_K}) - \frac{1}{2}\bar{\mathbf{y}}^T \frac{\partial \Delta}{\partial \Theta_K}\bar{\mathbf{y}} - \frac{1}{2}tr(\frac{\partial \Delta}{\partial \Theta_K}\Sigma_\mathbf{y})$$

$$\frac{\partial F}{\partial \Theta_T} = \frac{1}{2}tr(\mathbf{Z}^{-1}) - \frac{1}{2}\bar{\mathbf{y}}^T\bar{\mathbf{y}} - \frac{1}{2}tr(\Sigma_\mathbf{y})$$

$$\frac{\partial F}{\partial \epsilon} \approx \sum_{i=1}^{n} \frac{1 - 2\Phi(t_i \cdot \bar{y}_i)}{\epsilon + (1 - 2\epsilon)\Phi(t_i \cdot \bar{y}_i)} \text{ where: } \bar{y}_i = \int_\mathbf{y} y_i q(\mathbf{y})$$

It is easy to show that the provided approximation of the derivative $\frac{\partial F}{\partial \epsilon}$ equals zero, when $\epsilon = \frac{k}{n}$, where $k$ is the number of labeled data points differing in sign from their posterior means. The EM procedure described here is susceptible to local minima and in a few cases might be too slow to converge. Especially, when the evidence curve is flat and the initial values are far from the optimum, we found that the EM algorithm provided very small steps, thus, taking a long time to converge.

Whenever we encountered this problem in the experiments, we used an approximate gradient search to find a good value of initial parameters for the EM algorithm. Essentially as the gradients of the evidence are hard to compute, they can be approximated by the gradients of the lower bound and can be used in any gradient ascent procedure.

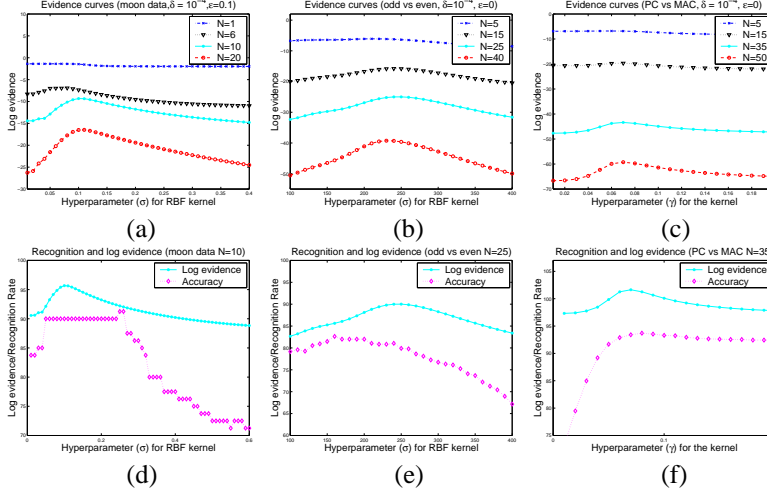

Figure 1: Evidence curves showing similar properties across different datasets (half-moon, odd vs even and PC vs MAC). The top row figures (a), (b) and (c) show the evidence curves for different amounts of labeled data per class. The bottom row figures (d), (e) and (f) show the correlation between recognition accuracy on unlabeled points and the evidence.

## 2.5 Classifying New Points

Since we compute a posterior distribution over the soft-labels of the labeled and unlabeled data points, classifying a new point is tricky. Note, that from the parameterization lemma for Gaussian Processes [1] it follows that given a prior distribution $p(\mathbf{y}|\mathbf{X}) \sim N(0, r(\Delta)^{-1})$, the mean of the posterior $p(\mathbf{y}|D)$ is a linear combination of the columns of $r(\Delta)^{-1}$. That is:

$$\bar{\mathbf{y}} = r(\Delta)^{-1}\mathbf{a} \quad \text{where,} \quad \mathbf{a} \in I\!\!R^{(n+m)\times 1}$$

Further, if the similarity matrix $K$ is a valid kernel matrix[1] then we can write the mean directly in terms of the linear combination of the columns of $K$:

$$\bar{\mathbf{y}} = KK^{-1}r(\Delta)^{-1}\mathbf{a} = K\mathbf{b} \tag{4}$$

Here, $\mathbf{b} = [b_1, .., b_{n+m}]^T$ is a column vector and is equal to $K^{-1}r(\Delta)^{-1}\mathbf{a}$. Thus, we have that $\bar{y}_i = \sum_{j=1}^{n+m} b_j \cdot K(\mathbf{x}_i, \mathbf{x}_j)$. This provides a natural extension of the framework to classify new points.

## 3 Experiments

We performed experiments to evaluate the three main contributions of this work: Bayesian hyperparameter learning, classification of unseen data points, and robustness with respect to noisy labels. For all the experiments we use the linear transformation $r(\lambda) = \lambda + \delta$ either on normalized Laplacian (EP-NL) or the combinatorial Laplacian (EP-CL). The experiments were performed on one synthetic (Figure 4(a)) and on three real-world datasets. Two real-world datasets were the handwritten digits and the newsgroup data from [7]. We evaluated the task of classifying odd vs even digits (15 labeled, 485 unlabeled and rest new

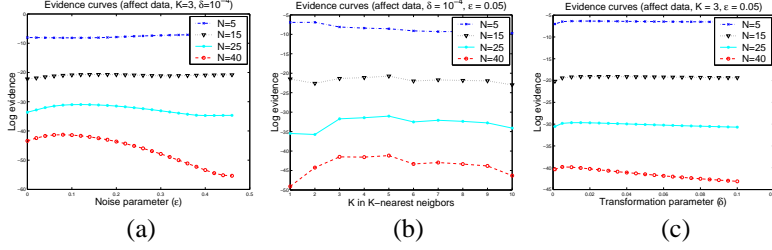

(a)           (b)           (c)

Figure 2: Evidence curves showing similar properties across different parameters of the model. The figures (a), (b) and (c) show the evidence curves for different amount of labeled data per class for the three different parameters in the model.

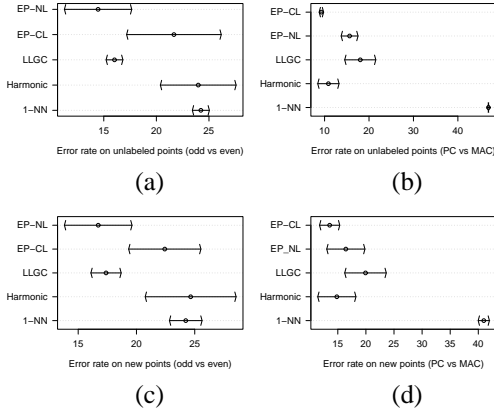

Figure 3: Error rates for different algorithms on digits (first column, (a) and (c)) and newsgroup dataset (second column (b) and (d)). The figures in the top row (a) and (b) show error rates on unlabeled points and the bottom row figures (c) and (d) on the new points. The results are averaged over 5 runs. Non-overlapping of error bars, the standard error scaled by 1.64, indicates 95% significance of the performance difference.

(unseen) points per class) and classifying PC vs MAC (5 labeled, 895 unlabeled and rest as new (unseen) points per class). An RBF kernel was used for handwritten digits, whereas kernel $K(\mathbf{x_i}, \mathbf{x_j}) = exp[-\frac{1}{\gamma}(1 - \frac{\mathbf{x_i}^T \mathbf{x_j}}{|\mathbf{x_i}||\mathbf{x_j}|})]$ was used on 10-NN graph to determine similarity. The third real-world dataset labels the level of interest (61 samples of high interest and 75 samples of low interest) of a child solving a puzzle on the computer. Each data point is a 19 dimensional real vector summarizing 8 seconds of activity from the face, posture and the puzzle. The labels in this database are suspected to be noisy because of human labeling. All the experiments on this data used K-nearest neighbor to determine the kernel matrix.

**Hyperparameter learning:** Figure 1 (a), (b) and (c) plots log evidence versus kernel parameters that determine the similarity graphs for the different datasets with varying size of the labeled set per class. The value of $\delta$ and $\epsilon$ were fixed to the values shown in the plots. Figure 2 (a), (b) and (c) plots the log evidence versus the noise parameter ($\epsilon$), the kernel parameter (k in k-NN) and the transformation parameter ($\delta$) for the affect dataset. First, we see that the evidence curves generated with very little data are flat and as the number of labeled data points increases we see the curves become peakier. When there is very little labeled data, there is not much information available for the evidence maximization framework to prefer one parameter value over the other. With more labeled data, the evidence curves become more informative. Figure 1 (d), (e) and (f) show the correlation between the evidence curves and the recognition rate on the unlabeled data and reveal that the recognition over the unlabeled data points is highly correlated with the evidence. Note that both of these effects are observed across all the datasets as well as all the different parameters, justifying evidence maximization for hyperparameter learning.

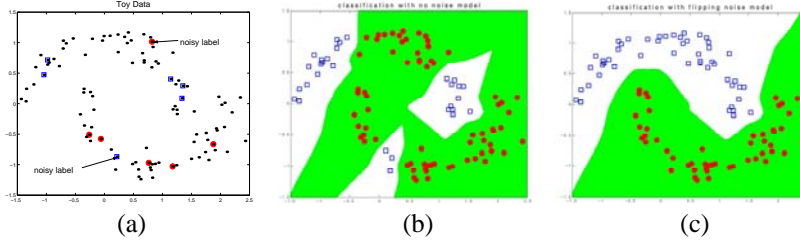

<div align="center">(a)            (b)           (c)</div>

Figure 4: Semi-supervised classification in presence of label noise. (a) Input data with label noise. Classification (b) without flipping noise model and with (c) flipping noise model.

**How good are the learnt parameters?** We performed experiments on the handwritten digits and on the newsgroup data and compared with 1-NN, LLGC and Harmonic approach. The kernel parameters for both LLGC and Harmonic were estimated using leave one out cross validation[2]. Note that both the approaches can be interpreted in terms of the new proposed Bayesian framework (see sec 2.1). We performed experiments with both the normalized (EP-NL) and the combinatorial Laplacian (EP-CL) with the proposed framework to classify the digits and the newsgroup data. The approximate gradient descent was first used to find an initial value of the kernel parameter for the EM algorithm. All three parameters were learnt and the top row in figure 3 shows the average error obtained for 5 different runs on the unlabeled points. On the task of classifying odd vs even the error rate for EP-NL was $14.46\pm4.4\%$, significantly outperforming the Harmonic ($23.98\pm4.9\%$) and 1-NN ($24.23\pm1.1\%$). Since the prior in EP-NL is determined using the normalized Laplacian and there is no label noise in the data, we expect EP-NL to at least work as well as LLGC ($16.02 \pm 1.1\%$). Similarly for the newsgroup dataset EP-CL ($9.28\pm0.7\%$) significantly beats LLGC ($18.03\pm3.5\%$) and 1-NN ($46.88\pm0.3\%$) and is better than Harmonic ($10.86\pm2.4\%$). Similar, results are obtained on new points as well. The unseen points were classified using eq. (4) and the nearest neighbor rule was used for LLGC and Harmonic.

**Handling label noise:** Figure 4(a) shows a synthetic dataset with noisy labels. We performed semi-supervised classification both with and without the likelihood model given in (3) and the EM algorithm was used to tune all the parameters including the noise ($\epsilon$). Besides modifying the spectrum of the Laplacian, the transformation parameter $\delta$ can also be considered as latent noise and provides a quadratic slack for the noisy labels [2]. The results are shown in figure 4 (b) and (c). The EM algorithm can correctly learn the noise parameter resulting in a perfect classification. The classification without the flipping model, even with the quadratic slack, cannot handle the noisy labels far from the decision boundary.

**Is there label noise in the data?** It was suspected that due to the manual labeling the affect dataset might have some label noise. To confirm this and as a sanity check, we first plotted evidence using *all* the available data. For all the semi-supervised methods in these experiments, we use 3-NN to induce the adjacency graph. Figure 5(a) shows the plot for the evidence against the noise parameter ($\epsilon$). From the figure, we see that the evidence peaks at $\epsilon = 0.05$ suggesting that the dataset has around 5% of labeling noise. Figure 5(b) shows comparisons with other semi-supervised (LLGC and SVM with graph kernel) and supervised methods (SVM with RBF kernel) for different sizes of the labeled dataset. Each point in the graph is the average error on 20 random splits of the data, where the error bars represent the standard error. EM was used to tune $\epsilon$ and $\delta$ in every run. We used the same transformation $r(\lambda) = \lambda + \delta$ on the graph kernel in the semi-supervised SVM. The hyperparameters in both the SVMs (including $\delta$ for the semi-supervised case) were estimated using leave one out. When the number of labeled points are small, both

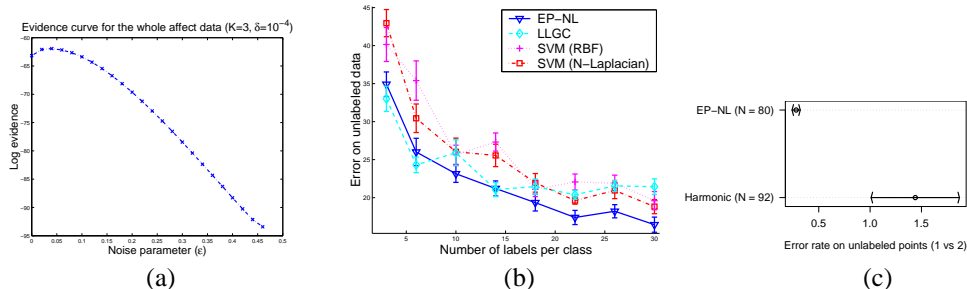

Figure 5: (a) Evidence vs noise parameter plotted using all the available data in the affect dataset. The maximum at $\epsilon = 0.05$ suggests that there is around 5% label noise in the data. (b) Performance comparison of the proposed approach with LLGC, SVM using graph kernel and the supervised SVM (RBF kernel) on the affect dataset which has label noise. The error bars represent the standard error. (c) Comparison of the proposed EM method for hyperparameter learning with the result reported in [7] using label entropy minimization. The plotted error bars represent the standard deviation.

LLGC and EP-NL perform similarly beating both the SVMs, but as the size of the labeled data increases we see a significant improvement of the proposed approach over the other methods. One of the reasons is when you have few labels the probability of the labeled set of points containing a noisy label is low. As the size of the labeled set increases the labeled data has more noisy labels. And, since LLGC has a Gaussian noise model, it cannot handle flipping noise well. As the number of labels increase, the evidence curve turns informative and EP-NL starts to learn the label noise correctly, outperforming the other Both the SVMs show competitive performance with more labels but still are worse than EP-NL. Finally, we also test the method on the task of classifying "1" vs "2" in the handwritten digits dataset. With 40 labeled examples per class (80 total labels and 1800 unlabeled), EP-NL obtained an average recognition accuracy of $99.72 \pm 0.04$% and figure 5(c) graphically shows the gain over the accuracy of $98.56 \pm 0.43$% reported in [7], where the hyperparameter were learnt by minimizing label entropy with 92 labeled and 2108 unlabeled examples.

## 4   Conclusion

We presented and evaluated a Bayesian framework for learning hyperparameters for graph-based semi-supervised classification. The results indicate that evidence maximization works well for learning hyperparameters, including the amount of label noise in the data.

## Footnotes

[1]The matrix $K$ is the adjacency matrix of the graph and depending upon the similarity criterion might not always be positive semi-definite. For example, discrete graphs induced using K-nearest neighbors might result in $K$ that is not positive semi-definite.

[2]Search space for $\sigma$ (odd vs even) was 100 to 400 with increments of 10 and for $\gamma$ (PC vs MAC) was 0.01 to 0.2 with increments of 0.1

### References

[1] Csato, L. (2002) Gaussian processes-iterative sparse approximation. *PhD Thesis*, Aston Univ.

[2] Kim, H. & Ghahramani, Z. (2004) The EM-EP algorithm for Gaussian process classification. *ECML*.

[3] Minka, T. P. (2001) Expectation propagation for approximate Bayesian inference. *UAI*.

[4] Opper, M. & Winther, O. (1999) Mean field methods for classification with Gaussian processes. *NIPS*.

[5] Smola, A. & Kondor, R. (2003) Kernels and regularization on graphs. *COLT*.

[6] Zhou et al. (2004) Learning with local and global consistency. *NIPS*.

[7] Zhu, X., Ghahramani, Z. & Lafferty, J. (2003) Semi-supervised learning using Gaussian fields and harmonic functions. *ICML*.

[8] Zhu, X., Kandola, J., Ghahramani, Z. & Lafferty, J. (2004) Nonparametric transforms of graph kernels for semi-supervised learning. *NIPS*.

[9] Zhu, X., Lafferty, J. & Ghahramani, Z. (2003) Semi-supervised learning: From Gaussian fields to Gaussian processes. CMU Tech Report:CMU-CS-03-175.
